# Subgraph Detection Using Eigenvector L1 Norms

**Benjamin A. Miller**
Lincoln Laboratory
Massachusetts Institute of Technology
Lexington, MA 02420
bamiller@ll.mit.edu

**Nadya T. Bliss**
Lincoln Laboratory
Massachusetts Institute of Technology
Lexington, MA 02420
nt@ll.mit.edu

**Patrick J. Wolfe**
Statistics and Information Sciences Laboratory
Harvard University
Cambridge, MA 02138
wolfe@stat.harvard.edu

## Abstract

When working with network datasets, the theoretical framework of detection theory for Euclidean vector spaces no longer applies. Nevertheless, it is desirable to determine the detectability of small, anomalous graphs embedded into background networks with known statistical properties. Casting the problem of subgraph detection in a signal processing context, this article provides a framework and empirical results that elucidate a "detection theory" for graph-valued data. Its focus is the detection of anomalies in unweighted, undirected graphs through $L_1$ properties of the eigenvectors of the graph's so-called modularity matrix. This metric is observed to have relatively low variance for certain categories of randomly-generated graphs, and to reveal the presence of an anomalous subgraph with reasonable reliability when the anomaly is not well-correlated with stronger portions of the background graph. An analysis of subgraphs in real network datasets confirms the efficacy of this approach.

## 1   Introduction

A graph $G = (V, E)$ denotes a collection of entities, represented by vertices $V$, along with some relationship between pairs, represented by edges $E$. Due to this ubiquitous structure, graphs are used in a variety of applications, including the natural sciences, social network analysis, and engineering. While this is a useful and popular way to represent data, it is difficult to analyze graphs in the traditional statistical framework of Euclidean vector spaces.

In this article we investigate the problem of detecting a small, dense subgraph embedded into an unweighted, undirected background. We use $L_1$ properties of the eigenvectors of the graph's modularity matrix to determine the presence of an anomaly, and show empirically that this technique has reasonable power to detect a dense subgraph where lower connectivity would be expected.

In Section 2 we briefly review previous work in the area of graph-based anomaly detection. In Section 3 we formalize our notion of graph anomalies, and describe our experimental regime. In Section 4 we give an overview of the modularity matrix and observe how its eigenstructure plays a role in anomaly detection. Sections 5 and 6 respectively detail subgraph detection results on simulated and actual network data, and in Section 7 we summarize and outline future research.

## 2  Related Work

The area of anomaly detection has, in recent years, expanded to graph-based data [1, 2]. The work of Noble and Cook [3] focuses on finding a subgraph that is dissimilar to a common substructure in the network. Eberle and Holder [4] extend this work using the minimum description length heuristic to determine a "normative pattern" in the graph from which the anomalous subgraph deviates, basing 3 detection algorithms on this property. This work, however, does not address the kind of anomaly we describe in Section 3; our background graphs may not have such a "normative pattern" that occurs over a significant amount of the graph. Research into anomaly detection in dynamic graphs by Priebe et al [5] uses the history of a node's neighborhood to detect anomalous behavior, but this is not directly applicable to our detection of anomalies in static graphs.

There has been research on the use of eigenvectors of matrices derived from the graphs of interest to detect anomalies. In [6] the angle of the principal eigenvector is tracked in a graph representing a computer system, and if the angle changes by more than some threshold, an anomaly is declared present. Network anomalies are also dealt with in [7], but here it is assumed that each node in the network has some highly correlated time-domain input. Since we are dealing with simple graphs, this method is not general enough for our purposes. Also, we want to determine the detectability of small anomalies that may not have a significant impact on one or two principal eigenvectors.

There has been a significant amount of work on community detection through spectral properties of graphs [8, 9, 10]. Here we specifically aim to detect small, dense communities by exploiting these same properties. The approach taken here is similar to that of [11], in which graph anomalies are detected by way of eigenspace projections. We here focus on smaller and more subtle subgraph anomalies that are not immediately revealed in a graph's principal components.

## 3  Graph Anomalies

As in [12, 11], we cast the problem of detecting a subgraph embedded in a background as one of detecting a signal in noise. Let $G_B = (V, E)$ denote the background graph; a network in which there exists no anomaly. This functions as the "noise" in our system. We then define the anomalous subgraph (the "signal") $G_S = (V_S, E_S)$ with $V_S \subset V$. The objective is then to evaluate the following binary hypothesis test; to decide between the null hypothesis $H_0$ and alternate hypothesis $H_1$:

$$\begin{cases} H_0 : & \text{The observed graph is "noise" } G_B \\ H_1 : & \text{The observed graph is "signal+noise" } G_B \cup G_S. \end{cases}$$

Here the union of the two graphs $G_B \cup G_S$ is defined as $G_B \cup G_S = (V, E \cup E_S)$.

In our simulations, we formulate our noise and signal graphs as follows. The background graph $G_B$ is created by a graph generator, such as those outlined in [13], with a certain set of parameters. We then create an anomalous "signal" graph $G_S$ to embed into the background. We select the vertex subset $V_S$ from the set of vertices in the network and embed $G_S$ into $G_B$ by updating the edge set to be $E \cup E_S$. We apply our detection algorithm to graphs with and without the embedding present to evaluate its performance.

## 4  The Modularity Matrix and its Eigenvectors

Newman's notion of the *modularity matrix* [8] associated with an unweighted, undirected graph $G$ is given by

$$B := A - \frac{1}{2|E|} K K^T. \tag{1}$$

Here $A = \{a_{ij}\}$ is the adjacency matrix of $G$, where $a_{ij}$ is 1 if there is an edge between vertex $i$ and vertex $j$ and is 0 otherwise; and $K$ is the degree vector of $G$, where the $i$th component of $K$ is the number of edges adjacent to vertex $i$. If we assume that edges from one vertex are equally likely to be shared with all other vertices, then the modularity matrix is the difference between the "actual" and "expected" number of edges between each pair of vertices. This is also very similar to

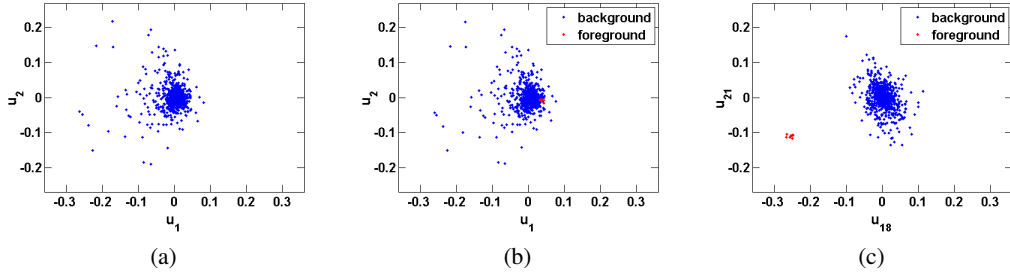

Figure 1: Scatterplots of an R-MAT generated graph projected into spaces spanned by two eigenvectors of its modularity matrix, with each point representing a vertex. The graph with no embedding (a) and with an embedded 8-vertex clique (b) look the same in the principal components, but the embedding is visible in the eigenvectors corresponding to the 18th and 21st largest eigenvalues (c).

the matrix used as an "observed-minus-expected" model in [14] to analyze the spectral properties of random graphs.

Since $B$ is real and symmetric, it admits the eigendecomposition $B = U\Lambda U^T$, where $U \in \mathbb{R}^{|V| \times |V|}$ is a matrix where each column is an eigenvector of $B$, and $\Lambda$ is a diagonal matrix of eigenvalues. We denote by $\lambda_i$, $1 \leq i \leq |V|$, the eigenvalues of $B$, where $\lambda_i \geq \lambda_{i+1}$ for all $i$, and by $u_i$ the unit-magnitude eigenvector corresponding to $\lambda_i$.

Newman analyzed the eigenvalues of the modularity matrix to determine if the graph can be split into two separate communities. As demonstrated in [11], analysis of the principal eigenvectors of $B$ can also reveal the presence of a small, tightly-connected component embedded in a large graph. This is done by projecting $B$ into the space of its two principal eigenvectors, calculating a Chi-squared test statistic, and comparing this to a threshold. Figure 1(a) demonstrates the projection of an R-MAT Kronecker graph [15] into the principal components of its modularity matrix.

Small graph anomalies, however, may not reveal themselves in this subspace. Figure 1(b) demonstrates an 8-vertex clique embedded into the same background graph. In the space of the two principal eigenvectors, the symmetry of the projection looks the same as in Figure 1(a). The foreground vertices are not at all separated from the background vertices, and the symmetry of the projection has not changed (implying no change in the test statistic). Considering only this subspace, the subgraph of interest cannot be detected reliably; its inward connectivity is not strong enough to stand out in the two principal eigenvectors.

The fact that the subgraph is absorbed into the background in the space of $u_1$ and $u_2$, however, does not imply that it is inseparable in general; only in the subspace with the highest variance. Borrowing language from signal processing, there may be another "channel" in which the anomalous signal subgraph can be separated from the background noise. There is in fact a space spanned by two eigenvectors in which the 8-vertex clique stands out: in the space of the $u_{18}$ and $u_{21}$, the two eigenvectors with the largest components in the rows corresponding to $V_S$, the subgraph is clearly separable from the background, as shown in Figure 1(c).

## 4.1 Eigenvector $L_1$ Norms

The subgraph detection technique we propose here is based on $L_1$ properties of the eigenvectors of the graph's modularity matrix, where the $L_1$ norm of a vector $x = \begin{bmatrix} x_1 & \cdots & x_N \end{bmatrix}^T$ is $\|x\|_1 := \sum_{i=1}^{N} |x_i|$. When a vector is closely aligned with a small number of axes, i.e., if $|x_i|$ is only large for a few values of $i$, then its $L_1$ norm will be smaller than that of a vector of the same magnitude where this is not the case. For example, if $x \in \mathbb{R}^{1024}$ has unit magnitude and only has nonzero components along two of the 1024 axes, then $\|x\|_1 \leq \sqrt{2}$. If it has a component of equal magnitude along all axes, then $\|x\|_1 = 32$. This property has been exploited in the past in a graph-theoretic setting, for finding maximal cliques [16, 17].

This property can also be useful when detecting anomalous clustering behavior. If there is a subgraph $G_S$ that is significantly different from its expectation, this will manifest itself in the modularity

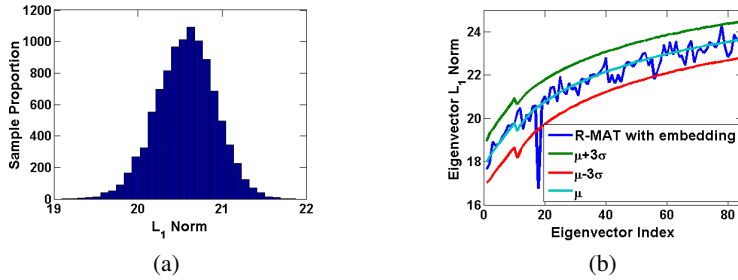

(a)  (b)

Figure 2: $L_1$ analysis of modularity matrix eigenvectors. Under the null model, $\|u_{18}\|$ has the distribution in (a). With an 8-vertex clique embedded, $\|u_{18}\|_1$ falls far from its average value, as shown in (b).

matrix as follows. The subgraph $G_S$ has a set of vertices $V_S$, which is associated with a set of indices corresponding to rows and columns of the adjacency matrix $A$. Consider the vector $x \in \{0,1\}^N$, where $x_i$ is 1 if $v_i \in V_S$ and $x_i = 0$ otherwise. For any $S \subseteq V$ and $v \in V$, let $d_S(v)$ denote the number of edges between the vertex $v$ and the vertex set $S$. Also, let $d_S(S') := \sum_{v \in S'} d_S(v)$ and $d(v) := d_V(v)$. We then have

$$\|Bx\|_2^2 = \sum_{v \in V} \left( d_{V_S}(v) - d(v)\frac{d(V_S)}{d(V)} \right)^2, \tag{2}$$

$$x^T Bx = d_{V_S}(V_S) - \frac{d^2(V_S)}{d(V)}, \tag{3}$$

and $\|x\|_2 = \sqrt{|V_S|}$. Note that $d(V) = 2|E|$. A natural interpretation of (2) is that $Bx$ represents the difference between the actual and expected connectivity to $V_S$ across the entire graph, and likewise (3) represents this difference *within* the subgraph. If $x$ is an eigenvector of $B$, then of course $x^T Bx/(\|Bx\|_2 \|x\|_2) = 1$. Letting each subgraph vertex have uniform internal and external degree, this ratio approaches 1 as $\sum_{v \notin V_S}(d_{V_S}(v) - d(v)d(V_S)/d(V))^2$ is dominated by $\sum_{v \in V_S}(d_{V_S}(v) - d(v)d(V_S)/d(V))^2$. This suggests that if $V_S$ is much more dense than a typical subset of background vertices, $x$ is likely to be well-correlated with an eigenvector of $B$. (This becomes more complicated when there are several eigenvalues that are approximately $d_{V_S}(V_S)/|V_S|$, but this typically occurs for smaller graphs than are of interest.) Newman made a similar observation: that the magnitude of a vertex's component in an eigenvector is related to the "strength" with which it is a member of the associated community. Thus if a small set of vertices forms a community, with few belonging to other communities, there will be an eigenvector well aligned with this set, and this implies that the $L_1$ norm of this eigenvector would be smaller than that of an eigenvector with a similar eigenvalue when there is no anomalously dense subgraph.

## 4.2 Null Model Characterization

To examine the $L_1$ behavior of the modularity matrix's eigenvectors, we performed the following experiment. Using the R-MAT generator we created 10,000 graphs with 1024 vertices, an average degree of 6 (the result being an average degree of about 12 since we make the graph undirected), and a probability matrix

$$P = \begin{bmatrix} 0.5 & 0.125 \\ 0.125 & 0.25 \end{bmatrix}.$$

For each graph, we compute the modularity matrix $B$ and its eigendecomposition. We then compute $\|u_i\|_1$ for each $i$ and store this value as part of our background statistics. Figure 2(a) demonstrates the distribution of $\|u_{18}\|_1$. The distribution has a slight left skew, but has a tight variance (a standard deviation of 0.35) and no large deviations from the mean under the null ($H_0$) model.

After compiling background data, we computed the mean and standard deviation of the $L_1$ norms for each $u_i$. Let $\mu_i$ be the average of $\|u_i\|_1$ and $\sigma_i$ be its standard deviation. Using the R-MAT graph with the embedded 8-vertex clique, we observed eigenvector $L_1$ norms as shown in Figure 2(b). In

the figure we plot $\|u_i\|_1$ as well as $\mu_i$, $\mu_i + 3\sigma_i$ and $\mu_i - 3\sigma_i$. The vast majority of eigenvectors have $L_1$ norms close to the mean for the associated index. There are very few cases with a deviation from the mean of greater than $3\sigma$. Note also that $\mu_i$ decreases with decreasing $i$. This suggests that the community formation inherent in the R-MAT generator creates components strongly associated with the eigenvectors with larger eigenvalues.

The one outlier is $u_{18}$, which has an $L_1$ norm that is over 10 standard deviations away from the mean. Note that $u_{18}$ is the horizontal axis in Figure 1(c), which by itself provides significant separation between the subgraph and the background. Simple $L_1$ analysis would certainly reveal the presence of this particular embedding.

## 5 Embedded Subgraph Detection

With the $L_1$ properties detailed in Section 4 in mind, we propose the following method to determine the presence of an embedding. Given a graph $G$, compute the eigendecomposition of its modularity matrix. For each eigenvector, calculate its $L_1$ norm, subtract its expected value (computed from the background statistics), and normalize by its standard deviation. If any of these modified $L_1$ norms is less than a certain threshold (since the embedding makes the $L_1$ norm smaller), $H_1$ is declared, and $H_0$ is declared otherwise. Pseudocode for this detection algorithm is provided in Algorithm 1.

---

**Algorithm 1** L1SUBGRAPHDETECTION

---

Input: Graph $G = (V, E)$, Integer $k$, Numbers $\ell_{1\text{MIN}}, \mu[1..k], \sigma[1..k]$
  $B \leftarrow \text{MODMAT}(G)$
  $U \leftarrow \text{EIGENVECTORS}(B, k)$  $\langle\langle k \text{ eigenvectors of } B \rangle\rangle$
  **for** $i \leftarrow 1$ to $k$ **do**
    $m[i] \leftarrow (\|u_i\|_1 - \mu[i])/\sigma[i]$
    **if** $m[i] < \ell_{1\text{MIN}}$ **then**
      return $H_1$  $\langle\langle$declare the presence of an embedding$\rangle\rangle$
    **end if**
  **end for**
  return $H_0$  $\langle\langle$no embedding found$\rangle\rangle$

---

We compute the eigenvectors of $B$ using `eigs` in MATLAB, which has running time $O(|E|kh + |V|k^2h + k^3h)$, where $h$ is the number of iterations required for `eigs` to converge [10]. While the modularity matrix is not sparse, it is the sum of a sparse matrix and a rank-one matrix, so we can still compute its eigenvalues efficiently, as mentioned in [8]. Computing the modified $L_1$ norms and comparing them to the threshold takes $O(|V|k)$ time, so the complexity is dominated by the eigendecomposition.

The signal subgraphs are created as follows. In all simulations in this section, $|V_S| = 8$. For each simulation, a subgraph density of 70%, 80%, 90% or 100% is chosen. For subraphs of this size and density, the method of [11] does not yield detection performance better than chance. The subgraph is created by, uniformly at random, selecting the chosen proportion of the $\binom{8}{2}$ possible edges. To determine where to embed the subgraph into the background, we find all vertices with at most 1, 3 or 5 edges and select 8 of these at random. The subgraph is then induced on these vertices.

For each density/external degree pair, we performed a 10,000-trial Monte Carlo simulation in which we create an R-MAT background with the same parameters as the null model, embed an anomalous subgraph as described above, and run Algorithm 1 with $k = 100$ to determine whether the embedding is detected. Figure 3 demonstrates detection performance in this experiment. In the receiver operating characteristic (ROC), changing the $L_1$ threshold ($\ell_{1\text{MIN}}$ in Algorithm 1) changes the position on the curve. Each curve corresponds to a different subgraph density. In Figure 3(a), each vertex of the subgraph has 1 edge adjacent to the background. In this case the subgraph connectivity is overwhelmingly inward, and the ROC curve reflects this. Also, the more dense subgraphs are more detectable. When the external degree is increased so that a subgraph vertex may have up to 3 edges adjacent to the background, we see a decline in detection performance as shown in Figure 3(b). Figure 3(c) demonstrates the additional decrease in detection performance when the external subgraph connectivity is increased again, to as much as 5 edges per vertex.

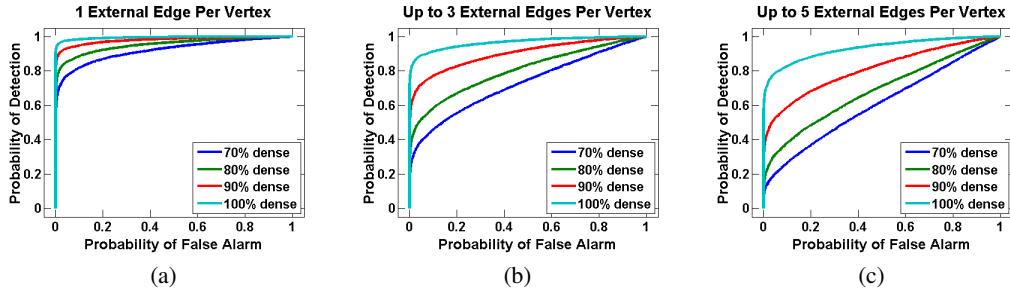

Figure 3: ROC curves for the detection of 8-vertex subgraphs in a 1024-vertex R-MAT background. Performance is shown for subgraphs of varying density when each foreground vertex is connected to the background by up to 1, 3 and 5 edges in (a), (b) and (c), respectively.

# 6 Subgraph Detection in Real-World Networks

To verify that we see similar properties in real graphs that we do in simulated ones, we analyzed five data sets available in the Stanford Network Analysis Package (SNAP) database [18]. Each network is made undirected before we perform our analysis. The data sets used here are the Epinions who-trusts-whom graph (Epinions, $|V|$ = 75,879, $|E|$ = 405,740) [19], the arXiv.org collaboration networks on astrophysics (AstroPh, $|V|$ = 18,722, $|E|$ = 198,050) and condensed matter (CondMat, $|V|$=23,133, $|E|$=93,439) [20], an autonomous system graph (asOregon, $|V|$=11,461, $|E|$=32,730) [21] and the Slashdot social network (Slashdot, $|V|$=82,168, $|E|$=504,230) [22]. For each graph, we compute the top 110 eigenvectors of the modularity matrix and the $L_1$ norm of each. Comparing each $L_1$ sequence to a "smoothed" (i.e., low-pass filtered) version, we choose the two eigenvectors that deviate the most from this trend, except in the case of Slashdot, where there is only one significant deviation.

Plots of the $L_1$ norms and scatterplots in the space of the two eigenvectors that deviate most are shown in Figure 4. The eigenvectors declared are highlighted. Note that, with the exception of the asOregon, we see as similar trend in these networks that we did in the R-MAT simulations, with the $L_1$ norms decreasing as the eigenvalues increase (the $L_1$ trend in asOregon is fairly flat). Also, with the exception of Slashdot, each dataset has a few eigenvectors with much smaller norms than those with similar eigenvalues (Slashdot decreases gradually, with one sharp drop at the maximum eigenvalue).

The subgraphs detected by $L_1$ analysis are presented in Table 1. Two subgraphs are chosen for each dataset, corresponding to the highlighted points in the scatterplots in Figure 4. For each subgraph we list the size (number of vertices), density (internal degree divided by the maximum number of edges), external degree, and the eigenvector that separates it from the background. The subgraphs are quite dense, at least 80% in each case.

To determine whether a detected subgraph is anomalous with respect to the rest of the graph, we sample the network and compare the sample graphs to the detected subgraphs in terms of density and external degree. For each detected subgraph, we take 1 million samples with the same number of vertices. Our sampling method consists of doing a random walk and adding all neighbors of each vertex in the path. We then count the number of samples with density above a certain threshold and external degree below another threshold. These thresholds are the parenthetical values in the 4th and 5th columns of Table 1. Note that the thresholds are set so that the detected subgraphs comfortably meet them. The 6th column lists the number of samples out of 1 million that satisfy both thresholds. In each case, far less than 1% of the samples meet the criteria. For the Slashdot dataset, no sample was nearly as dense as the two subgraphs we selected by thresholding along the principal eigenvector. After removing samples that are predominantly correlated with the selected eigenvectors, we get the parenthetical values in the same column. In most cases, all of the samples meeting the thresholds are correlated with the detected eigenvectors. Upon further inspection, those remaining are either correlated with another eigenvector that deviates from the overall $L_1$ trend, or correlated with *multiple* eigenvectors, as we discuss in the next section.

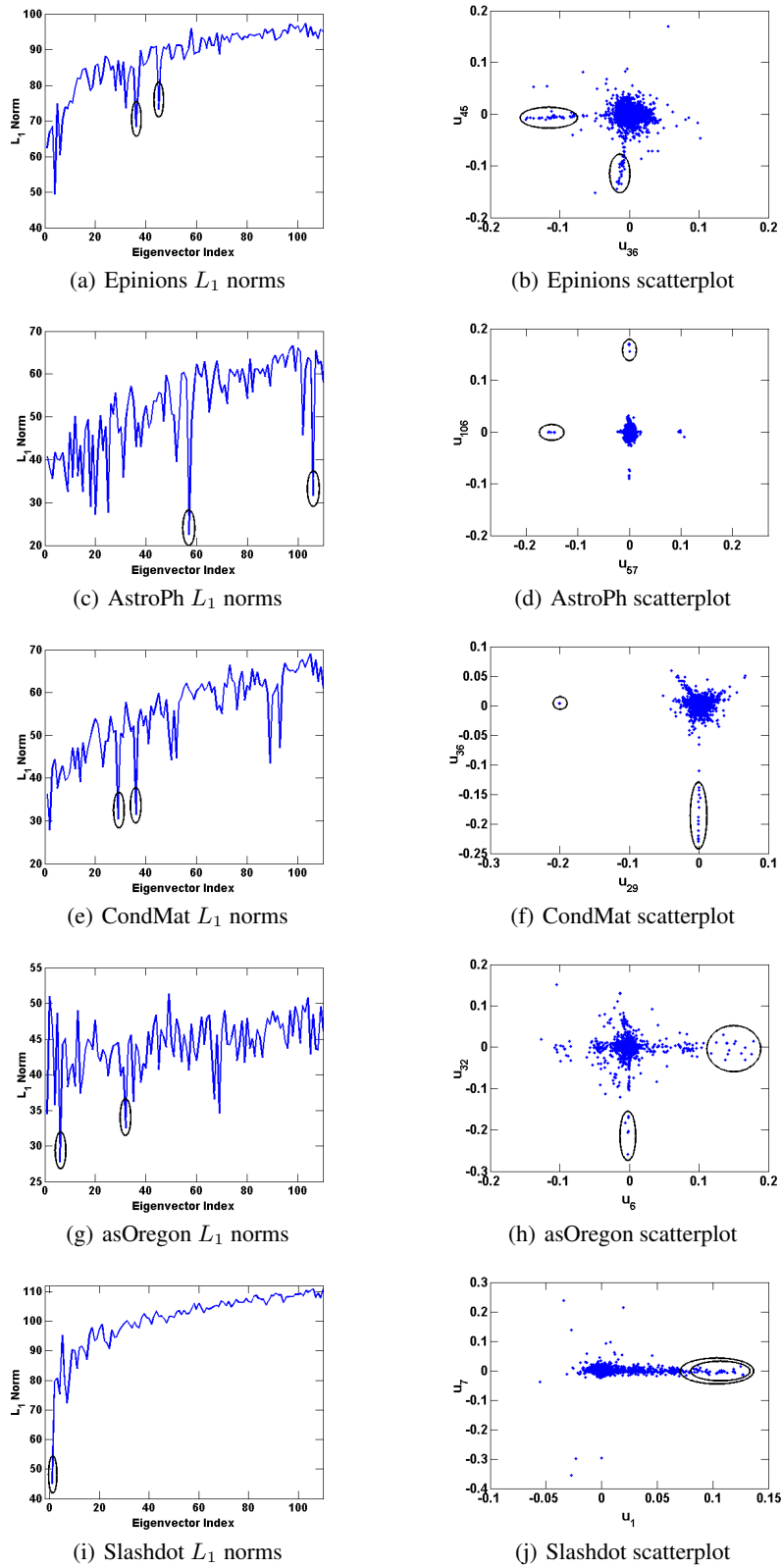

Figure 4: Eigenvector $L_1$ norms in real-world network data (left column), and scatterplots of the projection into the subspace defined by the indicated eigenvectors (right column).

| dataset | eigenvector | subgraph size | subgraph (sample) density | subgraph (sample) external degree | # samples that meet threshold |
|---|---|---|---|---|---|
| Epinions | $u_{36}$ | 34 | 80% (70%) | 721 (1000) | 46 (0) |
| Epinions | $u_{45}$ | 27 | 83% (75%) | 869 (1200) | 261 (6) |
| AstroPh | $u_{57}$ | 30 | 100% (90%) | 93 (125) | 853 (0) |
| AstroPh | $u_{106}$ | 24 | 100% (90%) | 73 (100) | 944 (0) |
| CondMat | $u_{29}$ | 19 | 100% (90%) | 2 (50) | 866 (0) |
| CondMat | $u_{36}$ | 20 | 83% (75%) | 70 (120) | 1596 (0) |
| asOregon | $u_6$ | 15 | 96% (85%) | 1089 (1500) | 23 (0) |
| asOregon | $u_{32}$ | 6 | 93% (80%) | 177 (200) | 762 (393) |
| Slashdot | $u_1 > 0.08$ | 36 | 95% (90%) | 10570 ($\infty$) | 0 (0) |
| Slashdot | $u_1 > 0.07$ | 51 | 89% (80%) | 12713 ($\infty$) | 0 (0) |

Table 1: Subgraphs detected by $L_1$ analysis, and a comparison with randomly-sampled subgraphs in the same network.

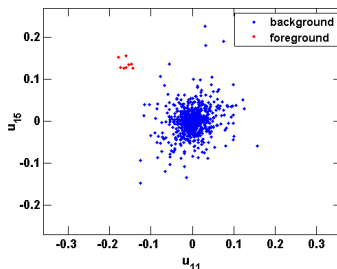

Figure 5: An 8-vertex clique that does not create an anomalously small $L_1$ norm in any eigenvector. The scatterplot looks similar to one in which the subgraph is detectable, but is rotated.

# 7 Conclusion

In this article we have demonstrated the efficacy of using eigenvector $L_1$ norms of a graph's modularity matrix to detect small, dense anomalous subgraphs embedded in a background. Casting the problem of subgraph detection in a signal processing context, we have provided the intuition behind the utility of this approach, and empirically demonstrated its effectiveness on a concrete example: detection of a dense subgraph embedded into a graph generated using known parameters. In real network data we see trends similar to those we see in simulation, and examine outliers to see what subgraphs are detected in real-world datasets.

Future research will include the expansion of this technique to reliably detect subgraphs that can be separated from the background in the space of a small number of eigenvectors, but not necessarily one. While the $L_1$ norm itself can indicate the presence of an embedding, it requires the subgraph to be highly correlated with a single eigenvector. Figure 5 demonstrates a case where considering multiple eigenvectors at once would likely improve detection performance. The scatterplot in this figure looks similar to the one in Figure 1(c), but is rotated such that the subgraph is equally aligned with the two eigenvectors into which the matrix has been projected. There is not significant separation in any one eigenvector, so it is difficult to detect using the method presented in this paper. Minimizing the $L_1$ norm with respect to rotation in the plane will likely make the test more powerful, but could prove computationally expensive. Other future work will focus on developing detectability bounds, the application of which would be useful when developing detection methods like the algorithm outlined here.

**Acknowledgments**

This work is sponsored by the Department of the Air Force under Air Force Contract FA8721-05-C-0002. Opinions, interpretations, conclusions and recommendations are those of the author and are not necessarily endorsed by the United States Government.

# References

[1] J. Sun, J. Qu, D. Chakrabarti, and C. Faloutsos, "Neighborhood formation and anomaly detection in bipartite graphs," in *Proc. IEEE Int'l. Conf. on Data Mining*, Nov. 2005.

[2] J. Sun, Y. Xie, H. Zhang, and C. Faloutsos, "Less is more: Compact matrix decomposition for large sparse graphs," in *Proc. SIAM Int'l. Conf. on Data Mining*, 2007.

[3] C. C. Noble and D. J. Cook, "Graph-based anomaly detection," in *Proc. ACM SIGKDD Int'l. Conf. on Knowledge Discovery and Data Mining*, pp. 631–636, 2003.

[4] W. Eberle and L. Holder, "Anomaly detection in data represented as graphs," *Intelligent Data Analysis*, vol. 11, pp. 663–689, December 2007.

[5] C. E. Priebe, J. M. Conroy, D. J. Marchette, and Y. Park, "Scan statistics on enron graphs," *Computational & Mathematical Organization Theory*, vol. 11, no. 3, pp. 229–247, 2005.

[6] T. Idé and H. Kashima, "Eigenspace-based anomaly detection in computer systems," in *Proc. KDD '04*, pp. 440–449, 2004.

[7] S. Hirose, K. Yamanishi, T. Nakata, and R. Fujimaki, "Network anomaly detection based on eigen equation compression," in *Proc. KDD '09*, pp. 1185–1193, 2009.

[8] M. E. J. Newman, "Finding community structure in networks using the eigenvectors of matrices," *Phys. Rev. E*, vol. 74, no. 3, 2006.

[9] J. Ruan and W. Zhang, "An efficient spectral algorithm for network community discovery and its applications to biological and social networks," in *Proc. IEEE Int'l Conf. on Data Mining*, pp. 643–648, 2007.

[10] S. White and P. Smyth, "A spectral clustering approach to finding communities in graphs," in *Proc. SIAM Data Mining Conf.*, 2005.

[11] B. A. Miller, N. T. Bliss, and P. J. Wolfe, "Toward signal processing theory for graphs and other non-Euclidean data," in *Proc. IEEE Int'l Conf. on Acoustics, Speech and Signal Processing*, pp. 5414–5417, 2010.

[12] T. Mifflin, "Detection theory on random graphs," in *Proc. Int'l Conf. on Information Fusion*, pp. 954–959, 2009.

[13] D. Chakrabarti and C. Faloutsos, "Graph mining: Laws, generators, and algorithms," *ACM Computing Surveys*, vol. 38, no. 1, 2006.

[14] F. Chung, L. Lu, and V. Vu, "The spectra of random graphs with given expected degrees," *Proc. of National Academy of Sciences of the USA*, vol. 100, no. 11, pp. 6313–6318, 2003.

[15] D. Chakrabarti, Y. Zhan, and C. Faloutsos, "R-MAT: A recursive model for graph mining," in *Proc. Fourth SIAM Int'l Conference on Data Mining*, vol. 6, pp. 442–446, 2004.

[16] T. S. Motzkin and E. G. Straus, "Maxima for graphs and a new proof of a theorem of Turán," *Canad. J. Math.*, vol. 17, pp. 533–540, 1965.

[17] C. Ding, T. Li, and M. I. Jordan, "Nonnegative matrix factorization for combinatorial optimization: Spectral clustering, graph matching, and clique finding," in *Proc. IEEE Int'l Conf. on Data Mining*, pp. 183–192, 2008.

[18] J. Leskovec, "Stanford network analysis package." http://snap.stanford.edu.

[19] M. Richardson, R. Agrawal, and P. Domingos, "Trust management for the semantic web," in *Proc. ISWC*, 2003.

[20] J. Leskovec, J. Kleinberg, and C. Faloutsos, "Graph evolution: Densification and shinking diameters," *ACM Trans. on Knowledge Discovery from Data*, vol. 1, no. 1, 2007.

[21] J. Leskovec, J. Kleinberg, and C. Faloutsos, "Graphs over time: Densification laws, shinking diameters and possible explanations," in *Proc. KDD '05*, 2005.

[22] J. Leskovec, K. Lang, A. Dasgupta, and M. Mahoney, "Community structure in large networks: Natural cluster sizes and the absence of large well-defined clusters." arXiv.org:0810.1355, 2008.

